# Separation of Music Signals by Harmonic Structure Modeling

**Yun-Gang Zhang**
Department of Automation
Tsinghua University
Beijing 100084, China
zyg00@mails.tsinghua.edu.cn

**Chang-Shui Zhang**
Department of Automation
Tsinghua University
Beijing 100084, China
zcs@mail.tsinghua.edu.cn

## Abstract

Separation of music signals is an interesting but difficult problem. It is helpful for many other music researches such as audio content analysis. In this paper, a new music signal separation method is proposed, which is based on harmonic structure modeling. The main idea of harmonic structure modeling is that the harmonic structure of a music signal is stable, so a music signal can be represented by a harmonic structure model. Accordingly, a corresponding separation algorithm is proposed. The main idea is to learn a harmonic structure model for each music signal in the mixture, and then separate signals by using these models to distinguish harmonic structures of different signals. Experimental results show that the algorithm can separate signals and obtain not only a very high Signal-to-Noise Ratio (SNR) but also a rather good subjective audio quality.

## 1 Introduction

Audio content analysis is an important area in music research. There are many open problems in this area, such as content based music retrieval and classification, Computational Auditory Scene Analysis (CASA), Multi-pitch Estimation, Automatic Transcription, Query by Humming, etc. [1, 2, 3, 4]. In all these problems, content extraction and representation is where the shoe pinches. In a song, the sounds of different instruments are mixed together, and it is difficult to parse the information of each instrument. Separation of sound sources in a mixture is a difficult problem and no reliable methods are available for the general case. However, music signals are so different from general signals. So, we try to find a way to separate music signals by utilizing the special character of music signals. After source separation, many audio content analysis problems will become much easier. In this paper, a music signal means a monophonic music signal performed by one instrument. A song is a mixture of several music signals and one or more singing voice signals.

As we know, music signals are more "ordered" than voice. The entropy of music is much more constant in time than that of speech [5]. More essentially, we found that an important character of a music signal is that its harmonic structure is stable. And the harmonic structures of music signals performed by different instruments are different. So, a harmonic structure model is built to represent a music signal. This model is the fundamental of the separation algorithm. In the separation algorithm, an extended multi-pitch estimation al-

gorithm is used to extract harmonic structures of all sources, and a clustering algorithm is used to calculate harmonic structure models. Then, signals are separated by using these models to distinguish harmonic structures of different signals.

There are many other signal separation methods, such as ICA [6]. General signal separation methods do not sufficiently utilize the special character of music signals. Gil-Jin and Te-Won proposed a probabilistic approach to single channel blind signal separation [7], which is based on exploiting the inherent time structure of sound sources by learning a priori sets of basis filters. In our approach, training sets are not required, and all information are directly learned from the mixture. Feng et al. applied FastICA to extract singing and accompaniment from a mixture [8]. Vanroose used ICA to remove music background from speech by subtracting ICA components with the lowest entropy [9]. Compared to these approaches, our method can separate each individual instrument sound, preserve the harmonic structure in the separated signals and obtain a good subjective audio quality. One of the most important contributions of our method is that it can significantly improve the accuracy of multi-pitch estimation. Compared to previous methods, our method learns models from the primary multi-pitch estimation results, and uses these models to improve the results. More importantly, pitches of different sources can be distinguished by these models. This advantage is significant for automatic transcription.

The rest of this paper is organized as follows: Harmonic structure modeling is detailed in Section two. The algorithm is described in section three. Experimental results are shown in section four. Finally, conclusion and discussions are given in section five.

## 2   Harmonic structure modeling for music signals

A monophonic music signal $s(t)$ can be represented by a sinusoidal model [10]:

$$s(t) = \sum_{r=1}^{R} A_r(t) \cos[\theta_r(t)] + e(t) \tag{1}$$

where $A_r(t)$ and $\theta_r(t) = \int_0^t 2\pi r f_0(\tau) d\tau$ are the instantaneous amplitude and phase of the $r^{th}$ harmonic, respectively, $R$ is the maximal harmonic number, $f_0(\tau)$ is the fundamental frequency at time $\tau$, $e(t)$ is the noise component.

We divide $s(t)$ into overlapped frames and calculate $f_0^l$ and $A_r^l$ by detecting peaks in the magnitude spectrum. $A_r^l = 0$, if there doesn't exist the $r^{th}$ harmonic. $l = 1, \ldots, L$ is the frame index. $f_0^l$ and $[A_1^l, \ldots, A_R^l]$ describe the position and amplitudes of harmonics. We normalize $A_r^l$ by multiplying a factor $\rho^l = C/A_1^l$ ( $C$ is an arbitrary constant) to eliminate the influence of the amplitude. We translate the amplitudes into a log scale, because the human ear has a roughly logarithmic sensitivity to signal intensity. Harmonic Structure Coefficient is then defined as equation (2). The timbre of a sound is mostly controlled by the number of harmonics and the ratio of their amplitudes, so $\mathbf{B}^l = [B_1^l, \ldots, B_R^l]$, which is free from the fundamental frequency and amplitude, exactly represents the timbre of a sound. In this paper, these coefficients are used to represent the harmonic structure of a sound. Average Harmonic Structure and Harmonic Structure Stability are defined as follows to model music signals and measure the stability of harmonic structures.

- Harmonic Structure $\mathbf{B}^l$, $B_i^l$ is Harmonic Structure Coefficient:
$$\mathbf{B}^l = [B_1^l, \ldots, B_R^l], B_i^l = \log(\rho^l A_i^l)/\log(\rho^l A_1^l), i = 1, \ldots, R \tag{2}$$

- Average Harmonic Structure (**AHS**): $\bar{\mathbf{B}} = \frac{1}{L} \sum_{l=1}^{L} \mathbf{B}^l$

- Harmonic Structure Stability (**HSS**):

$$HSS = \frac{1}{R} \cdot \frac{1}{L} \sum_{l=1}^{L} \left\| \mathbf{B}^l - \bar{\mathbf{B}} \right\|^2 = \frac{1}{RL} \sum_{r=1}^{R} \sum_{l=1}^{L} (B_r^l - \bar{B}_r)^2 \qquad (3)$$

AHS and HSS are the mean and variance of $\mathbf{B}^l$. Since timbres of most instruments are stable, $\mathbf{B}^l$ varies little in different frames in a music signal and AHS is a good model to represent music signals. On the contrary, $\mathbf{B}^l$ varies much in a voice signal and the corresponding HSS is much bigger than that of a music signal. See figure 1.

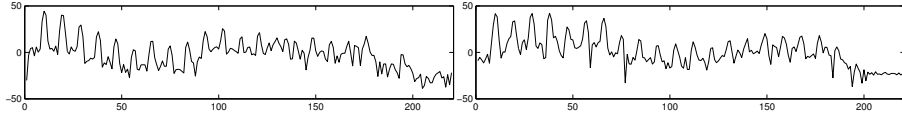

(a) Spectra in different frames of a voice signal. The number of harmonics (significant peaks in the spectrum) and their amplitude ratios are totally different.

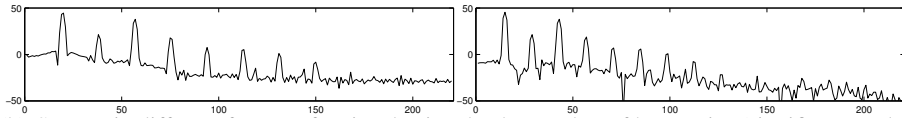

(b) Spectra in different frames of a piccolo signal. The number of harmonics (significant peaks in the spectrum) and their amplitude ratios are almost the same.

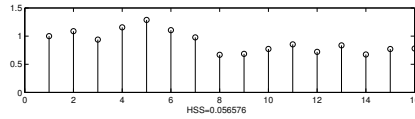 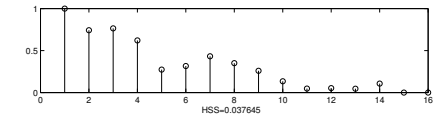

(c) The AHS and HSS of a oboe signal    (d) The AHS and HSS of a SopSax signal

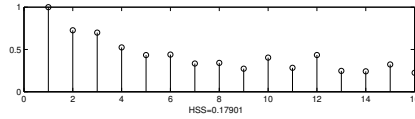 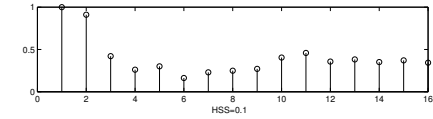

(e) The AHS and HSS of a male singing voice  (f) The AHS and HSS of a female singing voice

Figure 1: Spectra, AHSs and HSSs of voice and music signals. In (c)-(f), x-axis is harmonic number, y-axis is the corresponding harmonic structure coefficient.

## 3 Separation algorithm based on harmonic structure modeling

Without loss of generality, suppose we have a signal mixture consisting of one voice and several music signals. The separation algorithm consists of four steps: preprocessing, extraction of harmonic structures, music AHSs analysis, separation of signals.

In preprocessing step, the mean and energy of the input signal are normalized. In the second step, the pitch estimation algorithm of Terhardt [11] is extended and used to extract harmonic structures. This algorithm is suitable for estimating both the fundamental frequency and all its harmonics. In Terhardt's algorithm, in each frame, all spectral peaks exceeding a given threshold are detected. The frequencies of these peaks are $[f_1, \ldots, f_K]$, $K$ is the number of peaks. For a fundamental frequency candidate $f$, count the number of $f_i$ which satisfies the following condition:

$$floor[(1 + d)f_i/f] \geq (1 - d)f_i/f \qquad (4)$$

$floor(x)$ denotes the greatest integer less than or equal to $x$. This condition means whether $r_i f \cdot (1 - d) \leq f_i \leq r_i f \cdot (1 + d)$. If the condition is fulfilled, $f_i$ is the frequency of the $r_i^{th}$ harmonic component when fundamental frequency is $f$. For each fundamental frequency candidate $f$, the coincidence number is calculated and $\hat{f}$ corresponding to the largest coincidence number is selected as the estimated fundamental frequency.

The original algorithm is extended in the following ways: Firstly, not all peaks exceeding the given threshold are detected, only the significant ones are selected by an edge detection procedure. This is very important for eliminating noise and achieving high performances in next steps. Secondly, not only the fundamental frequency but also all its harmonics are extracted, then **B** can be calculated. Thirdly, the original optimality criterion is to select $\hat{f}$ corresponding to the largest coincidence number. This criterion is not stable when the signal is polyphonic, because harmonic components of different sources may influence each other. A new optimality criterion is define as follows ($n$ is the coincidence number):

$$d = \frac{1}{n} \sum_{i=1, f_i \text{ coincident with } f}^{K} \frac{|r_i - f_i/f|}{r_i} \qquad (5)$$

$\hat{f}$ corresponding to the smallest $d$ is the estimated fundamental frequency. The new criterion measures the precision of coincidence. For each fundamental frequency, harmonic components of the same source are more probably to have a high coincidence precision than those of a different source. So, the new criterion is helpful for separation of harmonic structures of different sources. Note that, the coincidence number is required to be larger than a threshold, such as 4-6. This requirement eliminates many errors. Finally, in the original algorithm, only one pitch was detected in each frame. Here, the sound is polyphonic. So, all pitches for which the corresponding $d$ is below a given threshold are extracted.

After harmonic structure extraction, a data set of harmonic structures is obtained. As the analysis in section two, in different frames, music harmonic structures of the same instrument are similar to each other and different from those of other instruments. So, in the data set all music harmonic structures form several high density clusters. Each cluster corresponds to an instrument. Voice harmonic structures scatter around like background noise, because the harmonic structure of the voice signal is not stable.

In the third step, NK algorithm [12] is used to learn music AHSs. NK algorithm is a clustering algorithm, which can cluster data on data sets consisting of clusters with different shapes, densities, sizes and even with some background noise. It can deal with high dimensional data sets. Actually, the harmonic structure data set is such a data set. Clusters of harmonic structures of different instruments have different densities. Voice harmonic structure are background noise. Each data point, a harmonic structure, has a high dimensionality (20 in our experiments). In NK algorithm, first find $K$ neighbors for each point and construct a neighborhood graph. Each point and its neighbors form a neighborhood. Then local PCA is used to calculate eigenvalues of a neighborhood. In a cluster, data points are close to each other and the neighborhood is small, so the corresponding eigenvalues are small. On the contrary, for a noise point, corresponding eigenvalues are much bigger. So noise points can be removed by eigenvalue analysis. After denoising, in the neighborhood graph, all points of a cluster are connected together by edges between neighbors. If two clusters are connected together, there must exist long edges between them. Then the eigenvalues of the corresponding neighborhoods are bigger than others. So all edges between clusters can be found and removed by eigenvalue analysis. Then data points are clustered correctly and AHSs can be obtained by calculate the mean of each cluster.

In the separation step, all harmonic structures of an instrument in all frames are extracted to reconstruct the corresponding music signals and then removed from the mixture. After removing all music signals, the rest of the mixture is the separated voice signal.

The procedure of music harmonic structure detection is detailed as follows. Given a music AHS $[\bar{B}_1, \ldots, \bar{B}_R]$ and a fundamental frequency candidate $f$, a music harmonic structure is predicted. $[f, 2f, \ldots, Rf]$ and $[\bar{B}_1, \ldots, \bar{B}_R]$ are its frequencies and harmonic structure coefficients. The closest peak in the magnitude spectrum for each predicted harmonic component is detected. Suppose $[f_1, \ldots, f_R]$ and $[B_1, \ldots, B_R]$ are the frequencies and harmonic structure coefficients of these peaks (measured peaks). Formula 6 is defined to calculate the distance between the predicted harmonic structure and the measured peaks.

$$
\begin{aligned}
D(f) = & \sum_{r=1, \bar{B}_r>0, B_r>0}^{R} \{\Delta f_r \cdot (rf)^{-p} + \frac{\bar{B}_r}{\bar{B}_{\max}} \times q\Delta f_r \cdot (rf)^{-p}\} \\
& + a \sum_{r=1, \bar{B}_r>0, B_r>0}^{R} (\frac{\bar{B}_r}{\bar{B}_{\max}})(\bar{B}_r - B_r)^2
\end{aligned}
\tag{6}
$$

The first part of $D$ is a modified version of Two-Way Mismatch measure defined by Maher and Beauchamp, which measures the frequency difference between predicted peaks and measured peaks [13], where $p$ and $q$ are parameters, and $\Delta f_r = |f_r - r \cdot f|$. The second part measures the shape difference between the two, $a$ is a normalization coefficient. Note that, only harmonic components with none-zero harmonic structure coefficients are considered. Let $\hat{f}$ indicate the fundamental frequency candidate corresponding to the smallest distance between the predicted peaks and the actual spectral peaks. If $D(\hat{f})$ is smaller than a threshold $T_d$, a music harmonic structure is detected. Otherwise there is no music harmonic structure in the frame. If a music harmonic structure is detected, the corresponding measured peaks in the spectrum are extracted, and the music signal is reconstructed by IFFT. Smoothing between frames is needed to eliminate errors and click noise between frames.

## 4    Experimental results

We have tested the performance of the proposed method on mixtures of different voice and music signals. The sample rate of the mixtures is 22.05kHz. Audio files for all the experiments are accessible at the website[1].

Figure 2 shows experimental results. In experiments 1 and 2, the mixed signals consist of one voice signal and one music signal. In experiment 3, the mixture consists of two music signals. In experiment 4, the mixture consists of one voice and two music signals. Table 1 shows SNR results. It can be seen that the mixtures are well separated into voice and music signals and very high SNRs are obtained in the separated signals. Experimental results show that music AHS is a good model for music signal representation and separation. There is another important fact that should be emphasized. In the separation procedure, music harmonic structures are detected by the music AHS model and separated from the mixture, and most of the time voice harmonic structures remain almost untouched. This procedure makes separated signals with a rather good subjective audio quality due to the good harmonic structure in the separated signals. Few existing methods can obtain such a good result because the harmonic structure is distorted in most of the existing methods.

It is difficult to compare our method with other methods, because they are so different. However, we compared our method with a speech enhancement method, because separation

Table 1: SNR results (DB): $snr_v$, $snr_{m1}$ and $snr_{m2}$ are the SNRs of voice and music signals in the mixed signal. $snr_e'$ is the SNR of speech enhancement result. $snr_v'$, $snr_{m1}'$ and $snr_{m2}'$ are the SNRs of the separated voice and music signals.

| | $snr_v$ | $snr_{m1}$ | $snr_{m2}$ | $snr_e'$ | $snr_v'$ | $snr_{m1}'$ | $snr_{m2}'$ | Total inc. |
|---|---|---|---|---|---|---|---|---|
| Experiment 1 | -7.9 | 7.9 | / | -6.0 | 6.7 | 10.8 | / | 17.5 |
| Experiment 2 | -5.2 | 5.2 | / | -1.5 | 6.6 | 10.0 | / | 16.6 |
| Experiment 3 | / | 1.6 | -1.6 | / | / | 9.3 | 7.1 | 16.4 |
| Experiment 4 | -10.0 | 0.7 | -2.2 | / | 2.8 | 8.6 | 6.3 | 29.2 |

of voice and music can be regarded as a speech enhancement problem by regarding music as background noise. Figure 2 (b), (d) give speech enhancement results obtained by a speech enhancement software which tries to estimate the spectrum of noise in the pause of speech and enhance the speech by spectral subtraction [14]. Detecting pauses in speech with music background and enhancing speech with fast music noise are both very difficult problems, so traditional speech enhancement techniques can't work here.

## 5  Conclusion and discussion

In this paper, a harmonic structure model is proposed to represent music signals and used to separate music signals. Experimental results show a good performance of this method.

The proposed method has many applications, such as multi-pitch estimation, audio content analysis, audio edit, speech enhancement with music background, etc.

Multi-pitch estimation is an important problem in music research. There are many existing methods, such as pitch perception model based methods, and probabilistic approaches [4, 15, 16, 17]. However, multi-pitch estimation is a very difficult problem and remains unsolved. Furthermore, it is difficult to distinguish pitches of different instruments in the mixture. In our algorithm, not only harmonic structures but also corresponding fundamental frequencies are extracted. So, the algorithm is also a new multi-pitch estimation method. It analyzes the primary multi-pitch estimation results and learns models to represent music signals and improve multi-pitch estimation results. More importantly, pitches of different sources can be distinguished by the AHS models. This advantage is significant for automatic transcription. Figure 2 (f) shows multi-pitch estimation results in experiment 3. It can be seen that, the multi-pitch estimation results are fairly good.

The proposed method is useful for melody extraction. As we know, in a mixed signal, multi-pitch estimation is a difficult problem. After separation, pitch estimation on the separated voice signal that contains melody becomes a monophonic pitch estimation problem, which can be done easily. The estimated pitch sequence represents the melody of the song. Then, many content base audio analysis tasks such as audio retrieval and classification become much easier and many midi based algorithms can be used on audio files.

There are still some limitations. Firstly, the proposed algorithm doesn't work for non-harmonic instruments, such as some drums. Some rhythm tracking algorithms can be used instead to separate drum sounds. Fortunately, most instrument sounds are harmonic. Secondly, for some instruments, the timbre in the onset is somewhat different from that in the stable duration. Also, different performing methods (pizz. or arco) produces different timbres. In these cases, the music harmonic structures of this instrument will form several clusters, not one. Then a GMM model instead of an average harmonic structure model (actually a point model) should be used to represent the music.

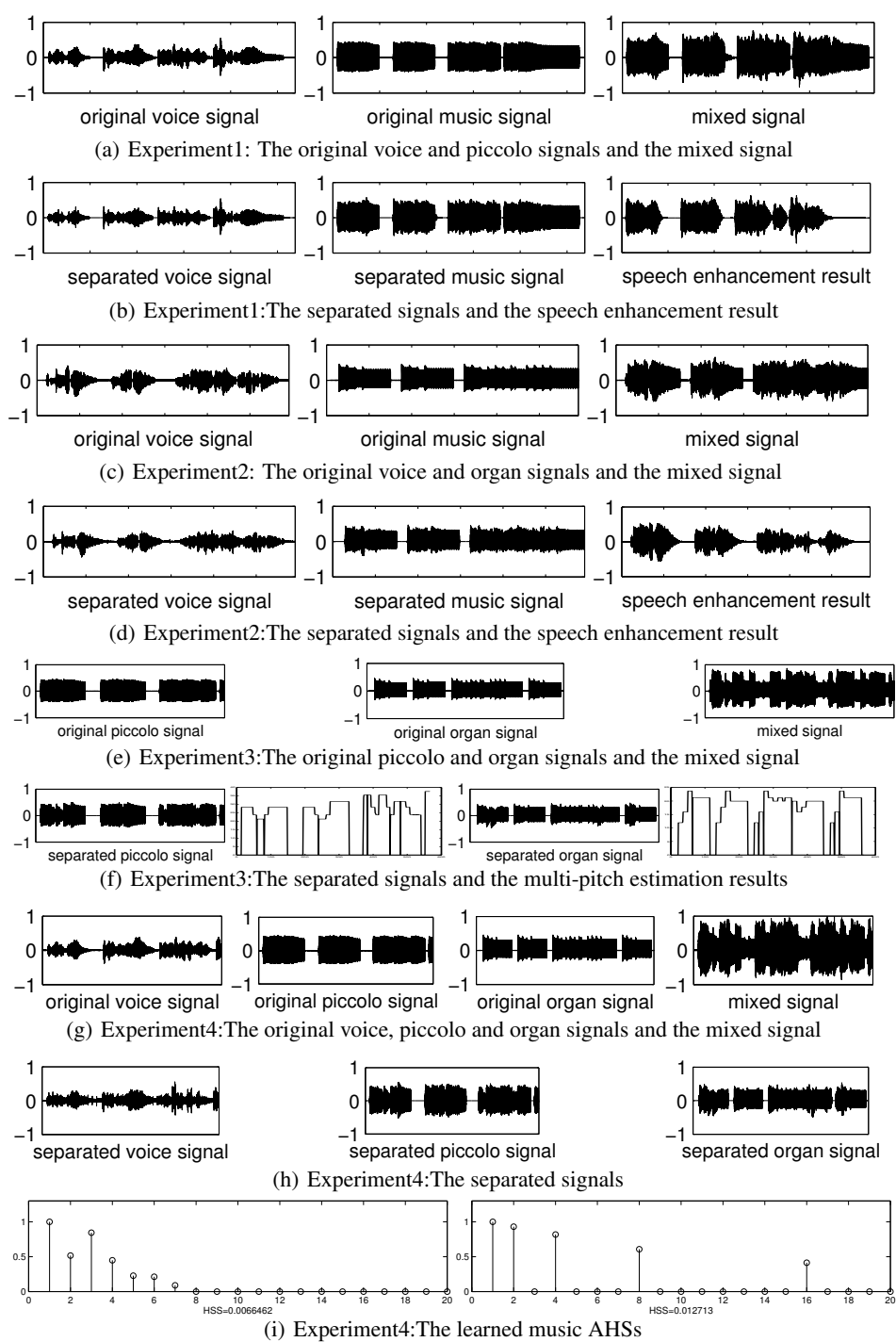

(a) Experiment1: The original voice and piccolo signals and the mixed signal

(b) Experiment1:The separated signals and the speech enhancement result

(c) Experiment2: The original voice and organ signals and the mixed signal

(d) Experiment2:The separated signals and the speech enhancement result

(e) Experiment3:The original piccolo and organ signals and the mixed signal

(f) Experiment3:The separated signals and the multi-pitch estimation results

(g) Experiment4:The original voice, piccolo and organ signals and the mixed signal

(h) Experiment4:The separated signals

(i) Experiment4:The learned music AHSs

Figure 2: Experimental results.

**Acknowledgments**

This work is supported by the project (60475001) of the National Natural Science Foundation of China.

## Footnotes

[1]http://www.au.tsinghua.edu.cn/szll/bodao/zhangchangshui/bigeye/member/zyghtm/experiments.htm

# References

[1] J. S. Downie, "Music information retrieval," *Annual Review of Information Science and Technology*, vol. 37, pp. 295–340, 2003.

[2] Roger Dannenberg, "Music understanding by computer," in *IAKTA/LIST International Workshop on Knowledge Technology in the Arts Proc.*, 1993, pp. 41–56.

[3] G. J. Brown and M. Cooke, "Computational auditory scene analysis," *Computer Speech and Language*, vol. 8, no. 4, pp. 297–336, 1994.

[4] M.Goto, "A robust predominant-f0 estimation method for real-time detection of melody and bass lines in cd recordings," in *IEEE International Conference on Acoustics, Speech, and Signal Processing (ICASSP2000)*, 2000, pp. 757–760.

[5] J. Pinquier, J. Rouas, and R. Andre-Obrecht, "Robust speech / music classification in audio documents," in *7th International Conference On Spoken Language Processing (ICSLP)*, 2002, pp. 2005–2008.

[6] P. Comon, "Independent component analysis, a new concept?," *Signal Processing*, vol. 36, pp. 287–314, 1994.

[7] Gil-Jin Jang and Te-Won Lee, "A probabilistic approach to single channel blind signal separation," in *Neural Information Processing Systems 15 (NIPS2002)*, 2003.

[8] Yazhong Feng, Yueting Zhuang, and Yunhe Pan, "Popular music retrieval by independent component analysis," in *ISMIR*, 2002, pp. 281–282.

[9] Peter Vanroose, "Blind source separation of speech and background music for improved speech recognition," in *The 24th Symposium on Information Theory*, May 2003, pp. 103–108.

[10] X. Serra, "Musical sound modeling with sinusoids plus noise," in *Musical Signal Processing*, C. Roads, S. Popea, A. Picialli, and G. De Poli, Eds. Swets & Zeitlinger Publishers, 1997.

[11] E. Terhardt, "Calculating virtual pitch," *Hearing Res.*, vol. 1, pp. 155–182, 1979.

[12] Yungang Zhang, Changshui Zhang, and Shijun Wang, "Clustering in knowledge embedded space," in *ECML*, 2003, pp. 480–491.

[13] R. C. Maher and J. W. Beauchamp, "Fundamental frequency estimation of musical signals using a two-way mismatch procedure," *Journal of the Acoustical Society of America*, vol. 95, no. 4, pp. 2254–2263, 1994.

[14] Serguei Koval, Mikhail Stolbov, and Mikhail Khitrov, "Broadband noise cancellation systems: new approach to working performance optimization," in *EUROSPEECH'99*, 1999, pp. 2607–2610.

[15] Anssi Klapuri, "Automatic transcription of music," M.S. thesis, Tampere University of Technology, Finland, 1998.

[16] Keerthi C. Nagaraj., "Toward automatic transcription - pitch tracking in polyphonic environment," Literature survey, Mar. 2003.

[17] Hirokazu Kameoka, Takuya Nishimoto, and Shigeki Sagayama, "Separation of harmonic structures based on tied gaussian mixture model and information criterion for concurrent sounds," in *IEEE International Conference on Acoustics, Speech, and Signal Processing (ICASSP04)*, 2004.
